# Skill Discovery in Continuous Reinforcement Learning Domains using Skill Chaining

**George Konidaris**
Computer Science Department
University of Massachusetts Amherst
Amherst MA 01003 USA
gdk@cs.umass.edu

**Andrew Barto**
Computer Science Department
University of Massachusetts Amherst
Amherst MA 01003 USA
barto@cs.umass.edu

## Abstract

We introduce a skill discovery method for reinforcement learning in continuous domains that constructs chains of skills leading to an end-of-task reward. We demonstrate experimentally that it creates appropriate skills and achieves performance benefits in a challenging continuous domain.

## 1 Introduction

Much recent research in reinforcement learning (RL) has focused on hierarchical RL methods [1] and in particular the *options* framework [2], which adds to the RL framework principled methods for planning and learning using high level-skills (called options). An important research goal is the development of methods by which an agent can discover useful new skills autonomously, and thereby construct its own high-level skill hierarchies. Although several methods exist for creating new options in discrete domains, none are immediately extensible to, or have been successfully applied in, continuous domains.

We introduce skill chaining, a skill discovery method for agents in continuous domains. Skill chaining produces chains of skills, each leading to one of a list of designated target events, where the list can simply contain the end-of-episode event or more sophisticated heuristic events (e.g., intrinsically interesting events [3]). The goal of each skill in the chain is to enable the agent to reach a state where its successor skill can be successfully executed. We demonstrate experimentally that skill chaining creates appropriate skills and achieves performance improvements in the Pinball domain.

## 2 Background and Related Work

An option, $o$, consists of three components [2]: an *option policy*, $\pi_o$, giving the probability of executing each action in each state in which the option is defined; an *initiation set* indicator function, $I_o$, which is 1 for states where the option can be executed and 0 elsewhere; and a *termination condition*, $\beta_o$, giving the probability of option execution terminating in states where it is defined. The options framework adds methods for planning and learning using options as temporally-extended actions to the standard RL framework based on the Markov decision process (MDP) framework [4]. Options can be added to an agent's action repertoire alongside its primitive actions, and the agent chooses when to execute them in the same way it chooses when to execute primitive actions.

Methods for creating new options must determine when to create an option and how to define its termination condition (skill discovery), how to expand its initiation set, and how to learn its policy. Given an *option reward function*, policy learning can be viewed as just another RL problem. Creation and termination are typically performed by the identification of option goal states, with an option created to reach one of its goal states and terminate when it does so. The initiation set is then the set

of states from which a goal state can be reached. In previous research, option goal states have been selected by a variety of methods, the most common relying on computing visit or reward statistics over individual states to identify useful subgoals [5, 6, 7, 8]. Graph-based methods [9, 10, 11] build a state graph and use its properties (e.g., local graph cuts [11]) to identify option goals. In domains with factored state spaces, the agent may create options to change infrequently changing variables [12, 13]. Finally, some methods extract options by exploiting commonalities in collections of policies over a single state space [14, 15, 16, 17]. All of these methods compute some statistic over individual states, in graphs derived from a set of state transitions, or rely on having state variables with finitely many values. These properties are unlikely to easily generalize to continuous spaces, where an agent may never see the same state twice.

We know of very little work on skill acquisition in continuous domains where the skills or action hierarchy are not designed in advance. Mugan and Kuipers [18] use learned qualitatively-discretized factored models of a continuous state space to derive options. This approach is restricted to domains where learning such a model is appropriate and feasible. In Neumann *et al.* [19], an agent learns to solve a complex task by sequencing motion templates. Both the template parameters and which templates to execute for each state are learned, although the agent's choices are constrained. However, the motion templates are parametrized policies designed specifically for the task.

The idea of arranging controllers so that executing one allows the next be executed is known in robotics as pre-image backchaining or sequential composition [20]. In such work the controllers and their pre-images (initiation sets) are typically given. Our work can be thought of as providing the means for learning control policies (and their regions of stability) that are suitable for sequential composition. The most recent relevant work in this line is by Tedrake [21], who builds a similar tree to ours in the model-based control setting, where the controllers are locally valid LQR controllers and their regions of stability (initiation sets) are computed using convex optimization. By contrast, our work does not require a model and may find superior (optimized) policies but does not provide formal guarantees.

## 3 Skill Discovery in Continuous Domains

In discrete domains, the primary reason for creating an option to reach a goal state is to make that state prominent in learning: a state that may once have been difficult to reach can now be reached using a single decision (to invoke the option). This effectively modifies the connectivity of the MDP by connecting the option's goal states to every state in its initiation set. Another reason for creating options is transfer: if options are learned in an appropriate space they can be used in later tasks to speed up learning. If the agent faces a sequence of tasks having the same state space, then options learned in it are portable [14, 15, 16, 17]; if it faces a sequence of tasks having different but related state spaces, then the options must be learned using features common to all the tasks [22].

In continuous domains, there is a further reason to create new options. An agent using function approximation to solve a problem must necessarily obtain an approximate solution. Creating new options that each have their own function approximator concentrated on a subset of the state space may result in better overall policies by freeing the primary value function from having to simultaneously represent the complexities of the individual option value functions. Thus, skill discovery offers an additional representational benefit in continuous domains. However, several difficulties that are absent or less apparent in discrete domains become important in continuous domains.

**Target regions.** Most existing skill discovery methods identify a single state as an option target. In continuous domains, where the agent may never see the same state twice, this must be generalized to a target region. However, simply defining the target region as a neighborhood about a point will not necessarily capture the goal of a skill. For example, many of the above methods generate target regions that are difficult to reach—a too-small neighborhood may make the target nearly impossible to reach; conversely, a too-large neighborhood may include regions that are not difficult to reach at all. Similarly, we cannot easily compute statistics over state space regions without first describing these regions, which is a nontrivial aspect of the problem.

**Initiation sets.** While in discrete domains it is common for an option's initiation set to expand arbitrarily as the agent learns a policy for successfully executing the option, this is not desirable in continuous domains. In discrete domains without function approximation a policy to reach a subgoal

can always be represented exactly; in continuous domains (or even discrete domains with function approximation), it may only be possible to represent such a policy locally. We are thus required to determine the extent of a new option's initiation set either analytically or through trial-and-error.

**Representation.** An option policy in both discrete and continuous domains should be able to consistently solve a simpler problem than the overall task using a simpler policy. A value table in a domain with a finite state set is a relatively simple data structure, and updates to it take constant time. Thus, in a discrete domain it is perfectly feasible to create a new value table for each learned option of the same dimension as the task value table. In continuous domains with many variables, however, value function approximation may require hundreds of even thousands of features to represent the overall task's value function, and updates are usually linear time. Therefore, "lightweight" options that use fewer features than needed to solve the overall problem are desirable in high-dimensional domains, or when we may wish to create many skills.

**Characterization.** Şimşek and Barto [8] characterize useful subgoals as those likely to lie on a solution path of the task the agent is facing. Options that are useful across a collection of problems should have goals that have high probability of falling on the solution paths of some of those problems (although not necessarily the one the agent is currently solving). In a discrete domain where the agent faces a finite number of tasks, one characterization of an option's usefulness may be obtained by treating the MDP as a graph and computing the likelihood that its goal lies on a solution path. Such a characterization is much more difficult in a continuous domain.

In the following section we develop an algorithm for skill discovery in continuous domains by addressing these challenges.

# 4  Skill Chaining

Since a useful option lies on a solution path, it seems natural to first create an option to reach the task's goal. The high likelihood that the option can only do so from a local neighborhood about this region suggests a follow-on step: create an option *to reach the states where the first option can be successfully executed.* This section describes skill chaining, a method that formalizes this intuition to create chains of options to reach a given target event by repeatedly creating options to reach options created earlier in the chain. First, we describe how to create an option given a target event.

## 4.1  Creating an Option to Trigger a Target Event

Given an episodic task defined over state space $S$ with reward function $R$, assume we are given a *goal trigger function* $T$ defined over $S$ that evaluates to 1 on states in the goal event and 0 otherwise. To create an option $o_T$ to trigger $T$, i.e., to reach a state on which $T$ evaluates to 1, we must define $o_T$'s termination condition, reward function, and initiation set.

For $o_T$'s termination condition we simply use $T$. We set $o_T$'s reward function to $R$ plus an option completion reward for triggering $T$. We can then use a standard RL algorithm to learn $o_T$'s policy, for example, using linear function approximation with a suitable set of basis functions to represent the option's value function. Obtaining $o_T$'s initiation set is more difficult because it should consist of the states from which executing $o_T$ succeeds in triggering $T$. We can treat this as a standard classification problem, using as positive training examples states in which $o_T$ has been executed and triggered $T$, and as negative training examples states in which it has been executed and failed to trigger $T$. A classifier suited to a potentially non-stationary classification problem with continuous features can be used to learn $o_T$'s initiation set.

## 4.2  Creating Skill Chains

Given an initial target event with trigger function $T_0$, which for the purposes of this discussion we consider to be the indicator function of the goal region of task, the agent creates a chain of skills as follows. First, the agent creates option $o_{T_0}$ to trigger $T_0$, learns a good policy for this option, and obtains a good estimate, $\hat{I}_{T_0}$, of its initiation set. We then add event $T_1 = \hat{I}_{T_0}$ to the list of target events, so that when the agent first enters $\hat{I}_{T_0}$, it creates a new option $o_{T_1}$ whose goal is to trigger $T_1$. That is, the new option's termination function is set to the indicator function $\hat{I}_{T_0}$, and its reward

function becomes the task's reward function plus an option completion reward for triggering $T_1$. Repeating this procedure results in a chain of skills leading from any state in which the agent may start to the task's goal region as depicted in Figure 1.

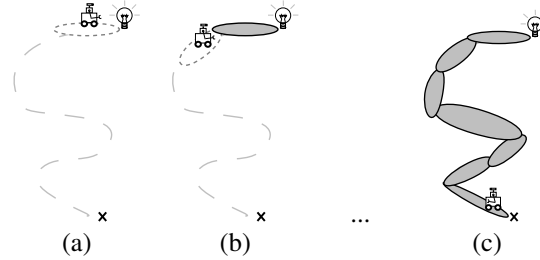

(a)          (b)     ...     (c)

Figure 1: An agent creates options using skill chaining. (a) First, the agent encounters a target event and creates an option to reach it. (b) Entering the initiation set of this first option triggers the creation of a second option whose target is the initiation set of the first option. (c) Finally, after many trajectories the agent has created a chain of options to reach the original target.

Note that although the options lie on a chain, the decision to execute each option is part of the agent's overall learning problem. Thus, they may not necessarily be executed sequentially; in particular, if an agent has learned a better policy for some parts of the chain, it may learn to skip some options.

## 4.3 Creating Skill Trees

The procedure above can create more general structures than chains. More than one option may be created to reach a target event if that event remains on the target event list after the first option is created to reach it. Each "child" option then creates its own chain, resulting in a *skill tree*, depicted in Figure 2. This will most likely occur when there are multiple solution trajectories (e.g., when the agent has multiple start states), or when noise or exploration create multiple segments along a solution path that cannot be covered by just one option.

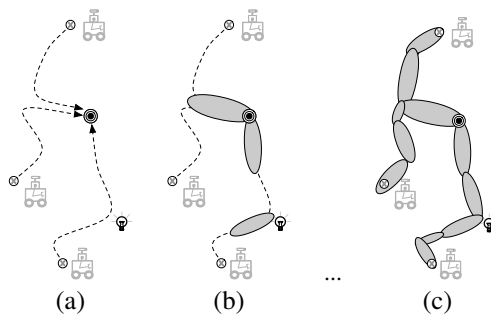

(a)          (b)     ...     (c)

Figure 2: (a) A skill chaining agent in an environment with multiple start states and two initial target events. (b) When the agent initially encounters target events it creates options to trigger them. (c) The initiation sets of these options then become target events, later triggering the creation of new options so that the agent eventually creates a skill tree covering all solution trajectories.

To control the branching factor of this tree, we need to place three further conditions on option creation. First, we do not create a new option when a target event is triggered from a state already in the initiation set of an option targeting that event. Second, we require that the initiation set of an option does not overlap that of its siblings or parents. (Note that although these conditions seem computationally expensive, they can be implemented using at most one execution of each initiation set classifier per visited state—which is required for action selection anyway). Finally, we may find it necessary to set a limit on the branching factor of the tree by removing a target event once it has some number of options targeting it.

## 4.4 More General Target Events

Although we have assumed that triggering the task's end-of-episode event is the only initial target event, we are free to start with *any* set of target events. We may thus include measures of novelty or other intrinsically motivating events [3] as triggers, events that are interesting for domain-specific reasons (e.g., physically meaningful events for a robot), or more general skill discovery techniques that can identify regions of interest before the goal is reached.

## 5 The Pinball Domain

Our experiments use two instances of the Pinball domain, shown in Figure 3.[1] The goal is to maneuver the small ball (which always starts in the same place in the first instance, and one of two places in the second) into the large red hole. The ball is dynamic (drag coefficient 0.995), so its state is described by four variables: $x$, $y$, $\dot{x}$ and $\dot{y}$. Collisions with obstacles are fully elastic and cause the ball to bounce, so rather than merely avoiding obstacles the agent may choose to use them to efficiently reach the hole. There are five primitive actions: incrementing or decrementing $\dot{x}$ or $\dot{y}$ by a small amount (which incurs a reward of $-5$ per action), or leaving them unchanged (which incurs a reward of $-1$ per action); reaching the goal obtains a reward of $10,000$.

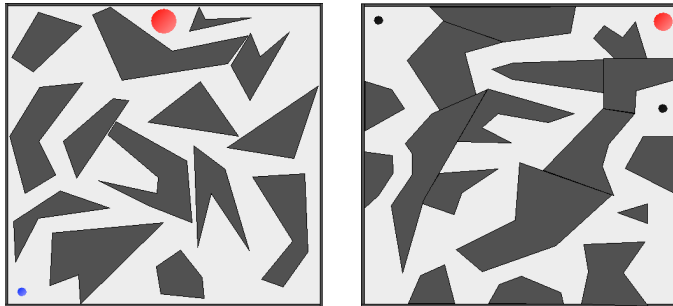

Figure 3: The two Pinball Domain instances used for our experiments.

The Pinball domain is an appropriate continuous domain for skill discovery because its dynamic aspects, sharp discontinuities, and extended dynamic control characteristics make it difficult for control and function approximation—much more difficult than a simple navigation task, or typical benchmarks like Acrobot. While a solution with a flat learning system is possible, there is scope for acquiring skills that could result in a better solution.

### 5.1 Implementation Details

To learn to solve the overall task for both standard and option-learning agents, we used Sarsa ($\gamma = 1, \epsilon = 0.01$) with linear function approximation, using a 4th-order Fourier basis [23] (625 basis functions per action) with $\alpha = 0.001$ for the first instance and a 5th-order Fourier basis (1296 basis functions per action) with $\alpha = 0.0005$ for the second (in both cases $\alpha$ was systematically varied and the best performing value used). Option policy learning was accomplished using Q-learning ($\alpha_o = 0.0005, \gamma = 1, \epsilon = 0.01$) with a 3rd-order Fourier basis (256 basis functions per action). Off-policy updates to an option for states outside its initiation set were ignored (because its policy does not need to be defined in those states), as were updates from unsuccessful on-policy trajectories (because their start states were then removed from the initiation set).

To initialize the option's policy before attempting to learn its initiation set, a newly created option was first allowed a "gestation period" of 10 episodes where it could not be executed and its policy was updated using only off-policy learning. After its gestation period, the option was added to the agent's action repertoire. For new option $o$, this requires expanding the overall action-value function $Q$ to include $o$ and assigning appropriate initial values to $Q(s, o)$. We therefore sampled the $Q$ values of transitions that triggered the option's target event during its gestation, and initialized $Q(s, o)$ to

[1]Java source code for Pinball can be downloaded at http://www-all.cs.umass.edu/~gdk/pinball

the maximum of these values. This reliably resulted in an optimistic but still fairly accurate initial value that encouraged the agent to execute the option.

Each option's initiation set was learned by a logistic regression classifier, initialized to be true everywhere, using 2nd order polynomial features, learning rate $\eta = 0.1$ and 100 sweeps per new data point. When the agent executed the option, states on trajectories that reached its goal within 250 steps were used as positive examples, and the start states of trajectories that did not were used as negative examples. We considered an option's initiation set learned well enough to be added to the list of target events when its weights changed on average less than $0.15$ per episode for two consecutive episodes. Since the Pinball domain has such strong discontinuities, to avoid over-generalization after this learning period we additionally constrained the initiation set to contain only points within a Euclidean distance of $0.1$ of a positive example. We used a maximum branching factor of 3.

## 6   Results

Figure 4(a) shows the performance (averaged over 100 runs) in the first Pinball instance for agents using a flat policy (without options) against agents employing skill chaining, and agents using given (pre-learned) options that were obtained using skill chaining over 250 episodes in the same task.

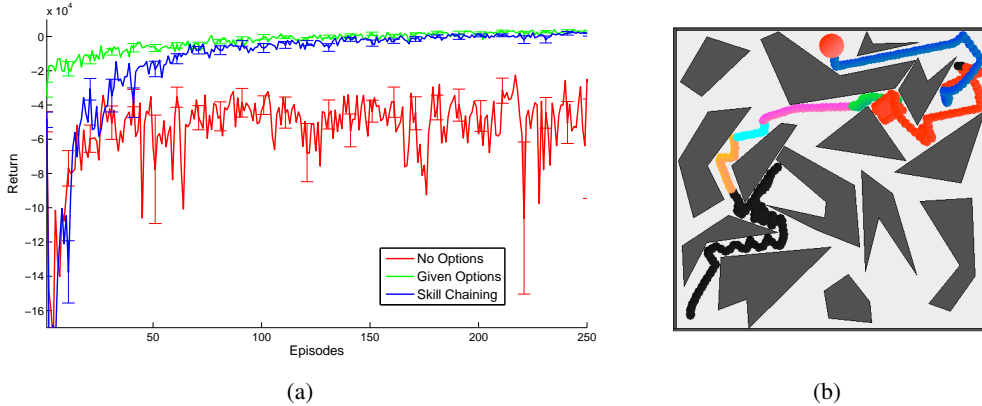

(a)                                                        (b)

Figure 4: (a) Performance in the first Pinball instance (averaged over 100 runs) for agents employing skill chaining, agents with given options, and agents without options. (b) A good example solution to the first Pinball instance, showing the acquired options executed along the sample trajectory in different colors. Primitive actions are in black.

Figure 4(a) shows that the skill chaining agents performed significantly better than flat agents by 50 episodes, and went on to obtain consistently good solutions by 250 episodes, whereas the flat agents did much worse and were less consistent. Agents that started with given options did very well initially—with an initial episode return far greater than the average solution eventually learned by agents without options—and proceeded quickly to the same quality of solution as the agents that discovered their options. This shows that the options themselves, and not the process of acquiring them, were responsible for the increase in performance.

Figure 4(b) shows a sample solution trajectory from an agent performing skill chaining in the first Pinball instance, with the options executed shown in different colors. The figure illustrates that this agent discovered options corresponding to simple, efficient policies covering segments of the sample trajectory. It also illustrates that in some places (in this case, the beginning of the trajectory) the agent learned to bypass a learned option—the black portions of the trajectory show where the agent employed primitive actions rather than a learned option. In some cases this occurred because poor policies were learned for those options. In this particular case, the presence of other options freed the overall policy (using a more complex function approximator) to represent the remaining trajectory segment better than could an option (with its less complex function approximator). Figure 5 shows the initiation sets and three sample trajectories from the options used in the trajectory shown in Figure 4(b). These learned initiation sets show that the discovered option policies are only locally valid, even though they are represented using Fourier basis functions, which have global support.

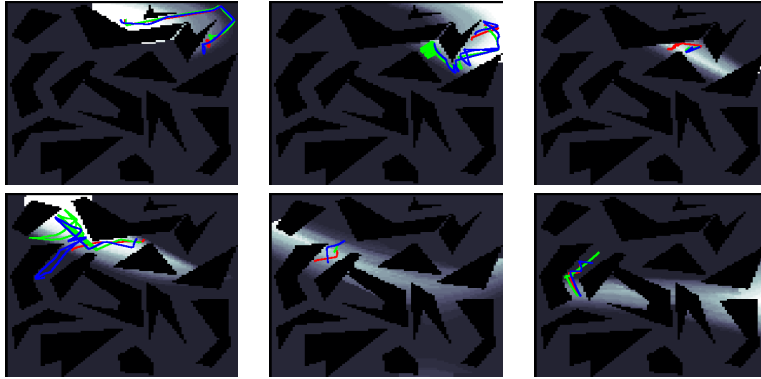

Figure 5: Initiation sets and sample policy trajectories for the options used in Figure 4(b). Each initiation set is shown using a density plot, with lightness increasing proportionally to the number of points in the set for a given $(x, y)$ coordinate, with $\dot{x}$ and $\dot{y}$ sampled over $\{-1, -\frac{1}{2}, 0, \frac{1}{2}, 1\}$.

Figures 6, 7 and 8 show similar results for the second Pinball instance, although Figure 6 shows a slight and transient initial penalty for skill chaining agents, before they go on to obtain far better and more consistent solutions than flat agents. The example trajectory in Figure 7 and initiation sets in Figure 8 show portions of a successfully formed skill tree.

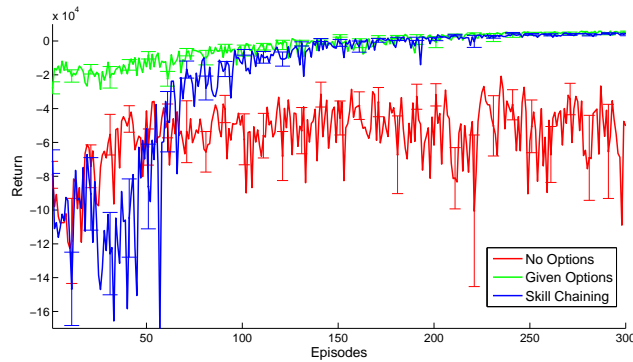

Figure 6: Performance in the second Pinball instance (averaged over 100 runs) for agents employing skill chaining, agents with given options, and agents without options.

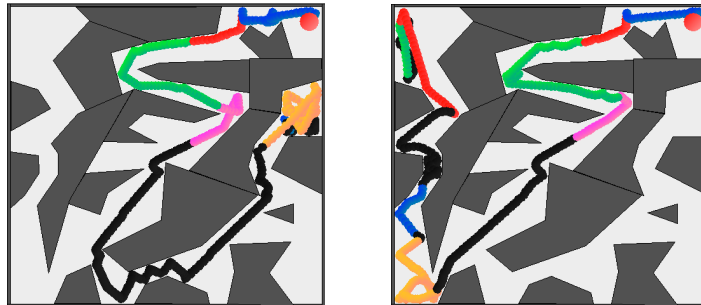

Figure 7: Good solutions to the second Pinball experimental domain, showing the acquired options executed along the sample trajectory in different colors. Primitive actions are shown in black.

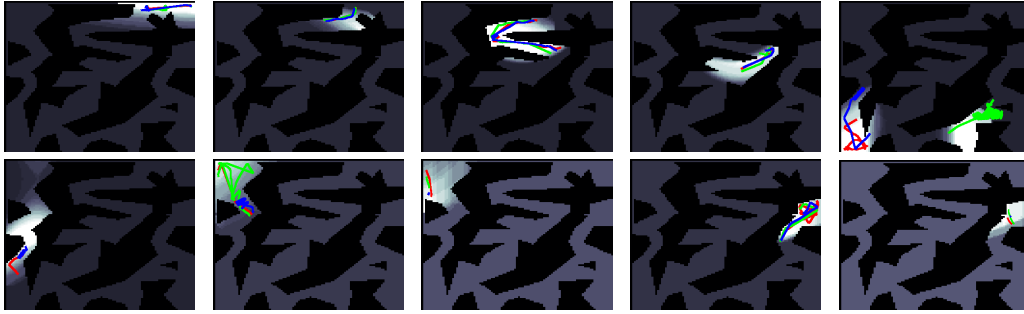

Figure 8: Initiation sets and sample trajectories for the options used in Figure 7.

# 7  Discussion and Conclusions

The performance gains demonstrated in the previous section show that skill chaining (at least using an end-of-episode target event) can significantly improve the performance of a RL agent in a challenging continuous domain, by breaking the solution into subtasks and learning lower-order option policies for each one.

Further benefits could be obtained by including more sophisticated initial target events: *any* indicator functions could be used in addition to the end-of-episode event. We expect that methods that identify regions likely to lie on the solution trajectory before a solution is found will result in the kinds of early performance gains sometimes seen in discrete skill discovery methods (e.g., [11]).

The primary benefit of skill chaining is that it reduces the burden of representing the task's value function, allowing each option to focus on representing its own local value function and thereby achieving a better overall solution. This implies that skill acquisition is best suited to high-dimensional problems where a single value function cannot be well represented using a feasible number of basis functions in reasonable time. In tasks where a good solution can be well represented using a low-order function approximator, we do not expect to see any benefits when using skill chaining.

Similar benefits may be obtainable using representation discovery methods [24], which construct basis functions to compactly represent complex value functions. We expect that such methods will prove most effective for extended control problems when combined with skill acquisition, where they can tailor a separate representation for each option rather than for the entire problem.

In this paper we used "lightweight" function approximators to represent option value functions. In domains such as robotics where the state space may contain thousands of state variables, we may require a more sophisticated approach that takes advantage of the notion that although the entire task may not be reducible to a feasibly sized state space, it is often possible to split it into subtasks that are. One such approach is *abstraction selection* [25, 26], where an agent uses sample trajectories (as obtained during gestation) to select an appropriate abstraction for a new option from a library of candidate abstractions, potentially resulting in a much easier learning problem.

We conjecture that the ability to discover new skills, and for each skill to employ its own abstraction, will prove a key advantage of hierarchical reinforcement learning as we try to scale up to extended control problems in high-dimensional spaces.

### Acknowledgments

We thank Jeff Johns, Özgür Şimşek and our reviewers for their helpful input. Andrew Barto was supported by the Air Force Office of Scientific Research under grant FA9550-08-1-0418.

# References

[1] A.G. Barto and S. Mahadevan. Recent advances in hierarchical reinforcement learning. *Discrete Event Systems*, 13:41–77, 2003. Special Issue on Reinforcement Learning.

[2] R.S. Sutton, D. Precup, and S.P. Singh. Between MDPs and semi-MDPs: A framework for temporal abstraction in reinforcement learning. *Artificial Intelligence*, 112(1-2):181–211, 1999.

[3] S. Singh, A.G. Barto, and N. Chentanez. Intrinsically motivated reinforcement learning. In *Proceedings of the 18th Annual Conference on Neural Information Processing Systems*, 2004.

[4] R.S. Sutton and A.G. Barto. *Reinforcement Learning: An Introduction*. MIT Press, Cambridge, MA, 1998.

[5] B.L. Digney. Learning hierarchical control structures for multiple tasks and changing environments. In *From Animals to Animats 5: Proceedings of the Fifth International Conference on Simulation of Adaptive Behavior*. MIT Press, 1998.

[6] A. McGovern and A.G. Barto. Automatic discovery of subgoals in reinforcement learning using diverse density. In *Proceedings of the 18th International Conference on Machine Learning*, pages 361–368, 2001.

[7] Ö. Şimşek and A.G. Barto. Using relative novelty to identify useful temporal abstractions in reinforcement learning. In *Proceedings of the 21st International Conference on Machine Learning*, pages 751–758, 2004.

[8] Ö. Şimşek and A.G. Barto. Skill characterization based on betweenness. In *Advances in Neural Information Processing Systems 22*, 2009.

[9] I. Menache, S. Mannor, and N. Shimkin. Q-cut—dynamic discovery of sub-goals in reinforcement learning. In *Proceedings of the 13th European Conference on Machine Learning*, pages 295–306, 2002.

[10] S. Mannor, I. Menache, A. Hoze, and U. Klein. Dynamic abstraction in reinforcement learning via clustering. In *Proceedings of the 21st International Conference on Machine Learning*, pages 560–567, 2004.

[11] Ö. Şimşek, A.P. Wolfe, and A.G. Barto. Identifying useful subgoals in reinforcement learning by local graph partitioning. In *Proceedings of the 22nd International Conference on Machine Learning*, 2005.

[12] B. Hengst. Discovering hierarchy in reinforcement learning with HEXQ. In *Proceedings of the 19th International Conference on Machine Learning*, pages 243–250, 2002.

[13] A. Jonsson and A.G. Barto. A causal approach to hierarchical decomposition of factored MDPs. In *Proceedings of the 22nd International Conference on Machine Learning*, 2005.

[14] S. Thrun and A. Schwartz. Finding structure in reinforcement learning. In *Advances in Neural Information Processing Systems*, volume 7, pages 385–392. The MIT Press, 1995.

[15] D.S. Bernstein. Reusing old policies to accelerate learning on new MDPs. Technical Report UM-CS-1999-026, Department of Computer Science, University of Massachusetts at Amherst, April 1999.

[16] T.J. Perkins and D. Precup. Using options for knowledge transfer in reinforcement learning. Technical Report UM-CS-1999-034, Department of Computer Science, University of Massachusetts Amherst, 1999.

[17] M. Pickett and A.G. Barto. Policyblocks: An algorithm for creating useful macro-actions in reinforcement learning. In *Proceedings of the 19th International Conference of Machine Learning*, pages 506–513, 2002.

[18] J. Mugan and B. Kuipers. Autonomously learning an action hierarchy using a learned qualitative state representation. In *Proceedings of the 21st International Joint Conference on Artificial Intelligence*, 2009.

[19] G. Neumann, W. Maass, and J. Peters. Learning complex motions by sequencing simpler motion templates. In *Proceedings of the 26th International Conference on Machine Learning*, 2009.

[20] R.R. Burridge, A.A. Rizzi, and D.E. Koditschek. Sequential composition of dynamically dextrous robot behaviors. *International Journal of Robotics Research*, 18(6):534–555, 1999.

[21] R. Tedrake. LQR-Trees: Feedback motion planning on sparse randomized trees. In *Proceedings of Robotics: Science and Systems*, 2009.

[22] G.D. Konidaris and A.G. Barto. Building portable options: Skill transfer in reinforcement learning. In *Proceedings of the 20th International Joint Conference on Artificial Intelligence*, 2007.

[23] G.D. Konidaris and S. Osentoski. Value function approximation in reinforcement learning using the Fourier basis. Technical Report UM-CS-2008-19, Department of Computer Science, University of Massachusetts Amherst, June 2008.

[24] S. Mahadevan. Learning representation and control in Markov Decision Processes: New frontiers. *Foundations and Trends in Machine Learning*, 1(4):403–565, 2009.

[25] G.D. Konidaris and A.G. Barto. Sensorimotor abstraction selection for efficient, autonomous robot skill acquisition. In *Proceedings of the 7th IEEE International Conference on Development and Learning*, 2008.

[26] G.D. Konidaris and A.G. Barto. Efficient skill learning using abstraction selection. In *Proceedings of the 21st International Joint Conference on Artificial Intelligence*, July 2009.

